# The Canonical Distortion Measure in Feature Space and 1-NN Classification

Jonathan Baxter* and Peter Bartlett
Department of Systems Engineering
Australian National University
Canberra 0200, Australia
{jon,bartlett}@syseng.anu.edu.au

## Abstract

We prove that the Canonical Distortion Measure (CDM) [2, 3] is the optimal distance measure to use for 1 nearest-neighbour (1-NN) classification, and show that it reduces to squared Euclidean distance in feature space for function classes that can be expressed as linear combinations of a fixed set of features. PAC-like bounds are given on the sample-complexity required to learn the CDM. An experiment is presented in which a neural network CDM was learnt for a Japanese OCR environment and then used to do 1-NN classification.

## 1 INTRODUCTION

Let $X$ be an input space, $P$ a distribution on $X$, $\mathcal{F}$ a class of functions mapping $X$ into $Y$ (called the "environment"), $Q$ a distribution on $\mathcal{F}$ and $\sigma$ a function $\sigma \colon Y \times Y \to [0, M]$. The *Canonical Distortion Measure* (CDM) between two inputs $x, x'$ is defined to be:

$$\rho(x, x') = \int_{\mathcal{F}} \sigma(f(x), f(x')) \, dQ(f). \tag{1}$$

Throughout this paper we will be considering real-valued functions and squared loss, so $Y = \mathbb{R}$ and $\sigma(y, y') := (y - y')^2$. The CDM was introduced in [2, 3], where it was analysed primarily from a vector quantization perspective. In particular, the CDM was proved to be the optimal distortion measure to use in vector quantization, in the sense of producing the best approximations to the functions in the environment $\mathcal{F}$. In [3] some experimental results were also presented (in a toy domain) showing how the CDM may be learnt.

The purpose of this paper is to investigate the utility of the CDM as a classification tool. In Section 2 we show how the CDM for a class of functions possessing a common feature

set reduces, via a change of variables, to squared Euclidean distance in feature space. A lemma is then given showing that the CDM is the optimal distance measure to use for 1-nearest-neighbour (1-NN) classification. Thus, for functions possessing a common feature set, optimal 1-NN classification is achieved by using squared Euclidean distance in feature space.

In general the CDM will be unknown, so in Section 4 we present a technique for learning the CDM by minimizing squared loss, and give PAC-like bounds on the sample-size required for good generalisation. In Section 5 we present some experimental results in which a set of features was learnt for a machine-printed Japanese OCR environment, and then squared Euclidean distance was used to do 1-NN classification in feature space. The experiments provide strong empirical support for the theoretical results in a difficult real-world application.

## 2   THE CDM IN FEATURE SPACE

Suppose each $f \in \mathcal{F}$ can be expressed as a linear combination of a fixed set of features $\Phi := (\phi_1, \ldots, \phi_k)$. That is, for all $f \in \mathcal{F}$, there exists $\mathbf{w} := (w_1, \ldots, w_k)$ such that $f = \mathbf{w} \cdot \Phi = \sum_{i=1}^{k} w_i \phi_i$. In this case the distribution $Q$ over the environment $\mathcal{F}$ is a distribution over the weight vectors $\mathbf{w}$. Measuring the distance between function values by $\sigma(y, y') := (y - y')^2$, the CDM (1) becomes:

$$\rho(x, x') = \int_{\mathbb{R}^k} \left[ \mathbf{w} \cdot \Phi(x) - \mathbf{w} \cdot \Phi(x') \right]^2 \, dQ(\mathbf{w}) = (\Phi(x) - \Phi(x'))W(\Phi(x) - \Phi(x'))'$$

(2)

where $W = \int_{\mathbf{w}} \mathbf{w}' \mathbf{w} \, dQ(\mathbf{w})$. is a $k \times k$ matrix. Making the change of variable $\Phi \rightarrow \Phi\sqrt{W}$, we have $\rho(x, x') = \|\Phi(x) - \Phi(x')\|^2$. Thus, the assumption that the functions in the environment can be expressed as linear combinations of a fixed set of features means that the CDM is simply squared Euclidean distance in a feature space related to the original by a linear transformation.

## 3   1-NN CLASSIFICATION AND THE CDM

Suppose the environment $\mathcal{F}$ consists of classifiers, *i.e.* $\{0, 1\}$-valued functions. Let $f$ be some function in $\mathcal{F}$ and $\mathbf{z} := (x_1, f(x_1)), \ldots, (x_n, f(x_n))$ a training set of examples of $f$. In 1-NN classification the classification of a novel $x$ is computed by $f(x^*)$ where $x^* = \mathrm{argmin}_{x_i} d(x, x_i))$, *i.e.* the classification of $x$ is the classification of the nearest training point to $x$ under some distance measure $d$. If both $f$ and $x$ are chosen at random, the expected misclassification error of the 1-NN scheme using $d$ and the training points $\mathbf{x} := (x_1, \ldots, x_n)$ is

$$\mathrm{er}(\mathbf{x}, d) := \mathbf{E}_{\mathcal{F}} \mathbf{E}_X \left[ f(x) - f(x^*) \right]^2, \qquad (3)$$

where $x^*$ is the nearest neighbour to $x$ from $\{x_1, \ldots, x_n\}$. The following lemma is now immediate from the definitions.

**Lemma 1.** *For all sequences* $\mathbf{x} = (x_1, \ldots, x_n)$, $\mathrm{er}(\mathbf{x}, d)$ *is minimized if $d$ is the CDM $\rho$.*

**Remarks.** Lemma 1 combined with the results of the last section shows that for function classes possessing a common feature set, optimal 1-NN classification is achieved by using squared Euclidean distance in feature space. In Section 5 some experimental results on Japanese OCR are presented supporting this conclusion.

The property of optimality of the CDM for 1-NN classification may not be stable to small perturbations. That is, if we learn an approximation $g$ to $\rho$, then even if $\mathbf{E}_{X \times X} (g(x, x') -$

$\rho(x, x'))^2$ is small it may not be the case that 1-NN classification using $g$ is also small. However, one can show that stability is maintained for classifier environments in which positive examples of different functions do not overlap significantly (as is the case for the Japanese OCR environment of Section 5, face recognition environments, speech recognition environments and so on). We are currently investigating the general conditions under which stability is maintained.

## 4 LEARNING THE CDM

For most environments encountered in practice (e.g speech recognition or image recognition), $\rho$ will be unknown. In this section it is shown how $\rho$ may be estimated or *learnt* using function approximation techniques (*e.g.* feedforward neural networks).

### 4.1 SAMPLING THE ENVIRONMENT

To learn the CDM $\rho$, the learner is provided with a class of functions (*e.g.* neural networks) $\mathcal{G}$ where each $g \in \mathcal{G}$ maps $X \times X \to [0, M]$. The goal of the learner is to find a $g$ such that the error between $g$ and the CDM $\rho$ is small. For the sake of argument this error will be measured by the expected squared loss:

$$\mathrm{er}_P(g) := \mathbf{E}_{X \times X} \left[ g(x, x') - \rho(x, x') \right]^2, \tag{4}$$

where the expectation is with respect to $P^2$.

Ordinarily the learner would be provided with training data in the form $(x, x', \rho(x, x'))$ and would use this data to minimize an empirical version of (4). However, $\rho$ is unknown so to generate data of this form $\rho$ must be estimated for each training pair $x, x'$. Hence to generate training sets for learning the CDM, both the distribution $Q$ over the environment $\mathcal{F}$ and the distribution $P$ over the input space $X$ must be sampled. So let $\mathbf{f} := (f_1, \ldots, f_m)$ be $m$ i.i.d. samples from $\mathcal{F}$ according to $Q$ and let $\mathbf{x} := (x_1, \ldots, x_n)$ be $n$ i.i.d. samples from $X$ according to $P$. For any pair $x_i, x_j$ an estimate of $\rho(x_i, x_j)$ is given by

$$\rho(x_i, x_j) := \frac{1}{m} \sum_{k=1}^{m} \sigma(f_k(x_i), f_k(x_j)). \tag{5}$$

This gives $n(n-1)/2$ training *triples*,

$$\{(x_i, x_j, \hat{\rho}(x_i, x_j)), 1 \leq i < j \leq n\},$$

which can be used as data to generate an empirical estimate of $\mathrm{er}_P(g)$:

$$\hat{\mathrm{er}}_{\mathbf{x}, \mathbf{f}}(g) := \frac{2}{n(n-1)} \sum_{1 \leq i < j \leq n} \left[ g(x_i, x_j) - \hat{\rho}(x_i, x_j) \right]^2. \tag{6}$$

Only $n(n-1)/2$ of the possible $n^2$ training triples are used because the functions $g \in \mathcal{G}$ are assumed to already be symmetric and to satisfy $g(x, x) = 0$ for all $x$ (if this is not the case then set $g'(x, x') := (g(x, x') + g(x', x))/2$ if $x \neq x'$ and $g'(x, x) = 0$ and use $\mathcal{G}' := \{g' : g \in \mathcal{G}\}$ instead).

In [3] an experiment was presented in which $\mathcal{G}$ was a neural network class and (6) was minimized directly by gradient descent. In Section 5 we present an alternative technique in which a set of features is first learnt for the environment and then an estimate of $\rho$ in feature space is constructed explicitly.

## 4.2 UNIFORM CONVERGENCE

We wish to ensure good generalisation from a $g$ minimizing $\hat{\mathrm{er}}_{\mathbf{x},\mathbf{f}}$, in the sense that (for small $\varepsilon,\delta$),

$$\Pr\left\{\mathbf{x},\mathbf{f}: \sup_{g\in\mathcal{G}}\left|\hat{\mathrm{er}}_{\mathbf{x},\mathbf{f}}(g)-\mathrm{er}_P(g)\right| > \varepsilon\right\} < \delta,$$

The following theorem shows that this occurs if both the number of functions $m$ and the number of input samples $n$ are sufficiently large. Some exotic (but nonetheless benign) measurability restrictions have been ignored in the statement of the theorem. In the statement of the theorem, $\mathcal{N}(\varepsilon,\mathcal{G})$ denotes the smallest $\varepsilon$-cover of $\mathcal{G}$ under the $L_1(P^2)$ norm, where $\{g_1,\ldots,g_N\}$ is an $\varepsilon$-cover of $\mathcal{G}$ if for all $g\in\mathcal{G}$ there exists $g_i$ such that $\|g_i-g\| \le \varepsilon$.

**Theorem 2.** *Assume the range of the functions in the environment $\mathcal{F}$ is no more than $[-\sqrt{B/2},\sqrt{B/2}]$ and in the class $\mathcal{G}$ (used to approximate the CDM) is no more than $[0,\sqrt{B}]$. For all $\varepsilon > 0$ and $0 < \delta \le 1$, if*

$$m \ge \frac{32B^4}{\varepsilon^2}\log\frac{4}{\delta} \tag{7}$$

*and*

$$n \ge \frac{512B^2}{\varepsilon^2}\left(\log\mathcal{N}(\varepsilon,\mathcal{G})+\log\frac{512B^2}{\varepsilon^2}+\log\frac{8}{\delta}\right) \tag{8}$$

*then*

$$\Pr\left\{\mathbf{x},\mathbf{f}: \sup_{g\in\mathcal{G}}\left|\hat{\mathrm{er}}_{\mathbf{x},\mathbf{f}}(g)-\mathrm{er}_P(g)\right| > \varepsilon\right\} < \delta. \tag{9}$$

*Proof.* For each $g\in\mathcal{G}$, define

$$\hat{\mathrm{er}}_{\mathbf{x}}(g) := \frac{2}{n(n-1)}\sum_{1\le i<j\le n}\left[g(x_i,x_j)-\rho(x_i,x_j)\right]^2. \tag{10}$$

If for any $\mathbf{x}=(x_1,\ldots,x_n)$,

$$\Pr\left\{\mathbf{f}: \sup_{g\in\mathcal{G}}|\hat{\mathrm{er}}_{\mathbf{x},\mathbf{f}}(g)-\hat{\mathrm{er}}_{\mathbf{x}}(g)| > \frac{\varepsilon}{2}\right\} \le \frac{\delta}{2}, \tag{11}$$

and

$$\Pr\left\{\mathbf{x}: \sup_{g\in\mathcal{G}}|\hat{\mathrm{er}}_{\mathbf{x}}(g)-\mathrm{er}_P(g)| > \frac{\varepsilon}{2}\right\} \le \frac{\delta}{2}, \tag{12}$$

then by the triangle inequality (9) will hold. We treat (11) and (12) separately.

**Equation** (11). To simplify the notation let $g_{ij}$, $\hat{\rho}_{ij}$ and $\rho_{ij}$ denote $g(x_i,x_j)$, $\hat{\rho}(x_i,x_j)$ and $\rho(x_i,x_j)$ respectively. Now,

$$\begin{aligned}
|\hat{\mathrm{er}}_{\mathbf{x},\mathbf{f}}(g)-\hat{\mathrm{er}}_{\mathbf{x}}(g)| &= \frac{2}{n(n-1)}\left|\sum_{1\le i<j\le n}(g_{ij}-\hat{\rho}_{ij})^2 - \sum_{1\le i<j\le n}(g_{ij}-\rho_{ij})^2\right| \\
&= \frac{2}{n(n-1)}\left|\sum_{1\le i<j\le n}(\rho_{ij}-\hat{\rho}_{ij})(2g_{ij}-\hat{\rho}_{ij}-\rho_{ij})\right| \\
&\le \frac{4B}{n(n-1)}\left|\sum_{1\le i<j\le n}(\rho_{ij}-\hat{\rho}_{ij})\right| \\
&= \left|\mathbf{E}_{\mathcal{F}}\mathbf{x}(f)-\frac{1}{m}\sum_{k=1}^{m}\mathbf{x}(f_k)\right|,
\end{aligned}$$

where $\mathbf{x}\colon \mathcal{F} \to [0, 4B^2]$ is defined by

$$\mathbf{x}(f) := \frac{4B}{n(n-1)} \sum_{1 \le i < j \le n} (f(x_i) - f(x_j))^2.$$

Thus,

$$\Pr\left\{ \mathbf{f}\colon \sup_{g \in \mathcal{G}} |\hat{\mathrm{er}}_{\mathbf{x},\mathbf{f}}(g) - \hat{\mathrm{er}}_{\mathbf{x}}(g)| > \frac{\varepsilon}{2} \right\} \le \Pr\left\{ \mathbf{f}\colon \left| \mathbf{E}_{\mathcal{F}}\mathbf{x}(f) - \frac{1}{m} \sum_{k=1}^{m} \mathbf{x}(f_k) \right| > \frac{\varepsilon}{2} \right\}$$

which is $\le 2\exp\left(-m\varepsilon^2/(32B^4)\right)$ by Hoeffding's inequality. Setting this less than $\delta/2$ gives the bound on $m$ in theorem 2.

**Equation** (12). Without loss of generality, suppose that $n$ is even. The trick here is to split the sum over all pairs $(x_i, x_j)$ (with $i < j$) appearing in the definition of $\hat{\mathrm{er}}_{\mathbf{x}}(g)$ into a double sum:

$$\hat{\mathrm{er}}_{\mathbf{x}}(g) = \frac{2}{n(n-1)} \sum_{1 \le i < j \le n} [g(x_i, x_j) - \rho(x_i, x_j)]^2$$

$$= \frac{1}{n-1} \sum_{i=1}^{n-1} \frac{2}{n} \sum_{j=1}^{n/2} \left[ g(x_{\sigma_i(j)}, x_{\sigma_i'(j)}) - \rho(x_{\sigma_i(j)}, x_{\sigma_i'(j)}) \right]^2,$$

where for each $i = 1, \ldots, n-1$, $\sigma_i$ and $\sigma_i'$ are permutations on $\{1, \ldots, n\}$ such that $\{\sigma_i(1), \ldots, \sigma_i(n/2)\} \cap \{\sigma_i'(1), \ldots, \sigma_i'(n/2)\}$ is empty. That there exist permutations with this property such that the sum can be broken up in this way can be proven easily by induction. Now, conditional on each $\sigma_i$, the $n/2$ pairs $\mathbf{x}_i := \{(x_{\sigma_i(j)}, x_{\sigma_i'(j)}), j = 1, \ldots, n/2\}$ are an i.i.d. sample from $X \times X$ according to $P^2$. So by standard results from real-valued function learning with squared loss [4]:

$$\Pr\left\{ \mathbf{x}_i\colon \sup_{g \in \mathcal{G}} \left| \frac{2}{n} \sum_{j=1}^{n/2} \left[ g(x_{\sigma_i(j)}, x_{\sigma_i'(j)}) - \rho(x_{\sigma_i(j)}, x_{\sigma_i'(j)}) \right]^2 - \mathrm{er}_P(g) \right| > \frac{\varepsilon}{2} \right\}$$

$$\le 4\mathcal{N}\left( \frac{\varepsilon}{48B^2}, \mathcal{G} \right) \exp\left( -\frac{n\varepsilon^2}{256B^2} \right).$$

Hence, by the union bound,

$$\Pr\left\{ \mathbf{x}\colon \sup_{g \in \mathcal{G}} |\hat{\mathrm{er}}_{\mathbf{x}}(g) - \mathrm{er}_P(g)| > \frac{\varepsilon}{2} \right\} \le 4(n-1)\mathcal{N}\left( \frac{\varepsilon}{48B^2}, \mathcal{G} \right) \exp\left( -\frac{n\varepsilon^2}{256B^2} \right).$$

Setting $n$ as in the statement of the theorem ensures this is less than $\delta/2$. $\qquad\square$

**Remark.** The bound on $m$ (the number of functions that need to be sampled from the environment) is independent of the complexity of the class $\mathcal{G}$. This should be contrasted with related bias learning (or equivalently, learning to learn) results [1] in which the number of functions does depend on the complexity. The heuristic explanation for this is that here we are only learning a distance function on the input space (the CDM), whereas in bias learning we are learning an entire hypothesis space that is appropriate for the environment. However, we shall see in the next section how for certain classes of problems the CDM can also be used to learn the functions in the environment. Hence in these cases learning the CDM is a more effective method of learning to learn.

## 5  EXPERIMENT: JAPANESE OCR

To verify the optimality of the CDM for 1-NN classification, and also to show how it can be learnt in a non-trivial domain (only a toy example was given in [3]), the

CDM was learnt for a Japanese OCR environment. Specifically, there were 3018 functions $f$ in the environment $\mathcal{F}$, each one a classifier for a different Kanji character. A database containing 90,918 segmented, machine-printed Kanji characters scanned from various sources was purchased from the CEDAR group at the State University of New York, Buffalo The quality of the images ranged from clean to very degraded (see `http://www.cedar.buffalo.edu/Databases/JOCR/`).

The main reason for choosing Japanese OCR rather than English OCR as a test-bed was the large number of distinct characters in Japanese. Recall from Theorem 2 that to get good generalisation from a learnt CDM, sufficiently many functions must be sampled from the environment. If the environment just consisted of English characters then it is likely that "sufficiently many" characters would mean *all* characters, and so it would be impossible to test the learnt CDM on novel characters not seen in training.

Instead of learning the CDM directly by minimizing (6), it was learnt implicitly by first learning a set of neural network features for the functions in the environment. The features were learnt using the method outlined in [1], which essentially involves learning a set of classifiers with a common final hidden layer. The features were learnt on 400 out of the 3000 classifiers in the environment, using 90% of the data in training and 10% in testing. Each resulting classifier was a linear combination of the neural network features. The average error of the classifiers was 2.85% on the test set (which is an accurate estimate as there were 9092 test examples).

Recall from Section 2 that if all $f \in \mathcal{F}$ can be expressed as $f = \mathbf{w} \cdot \Phi$ for a fixed feature set $\Phi$, then the CDM reduces to $\rho(x, x') = (\Phi(x) - \Phi(x'))W(\Phi(x) - \Phi(x'))'$ where $W = \int_{\mathbf{w}} \mathbf{w}'\mathbf{w} \, dQ(\mathbf{w})$. The result of the learning procedure above is a set of features $\hat{\Phi}$ and 400 weight vectors $\mathbf{w}_1, \ldots, \mathbf{w}_{400}$, such that for each of the character classifiers $f_i$ used in training, $f_i \simeq \mathbf{w}_i \cdot \hat{\Phi}$. Thus, $g(x, x') := (\hat{\Phi}(x) - \hat{\Phi}(x'))W(\hat{\Phi}(x) - \hat{\Phi}(x'))'$ is an empirical estimate of the true CDM, where $W := \sum_{i=1}^{400} \mathbf{w}_i'\mathbf{w}_i$. With a linear change of variable $\hat{\Phi} \to \hat{\Phi}\sqrt{W}$, $g$ becomes $g(x, x') = \|\hat{\Phi}(x) - \hat{\Phi}(x')\|^2$. This $g$ was used to do 1-NN classification on the test examples in two different experiments.

In the first experiment, all testing and training examples that were not an example of one of the 400 training characters were lumped into an extra category for the purpose of classification. All test examples were then given the label of their nearest neighbour in the training set under $g$ (*i.e.* , initially all training examples were mapped into feature space to give $\{\hat{\Phi}(x_1), \ldots, \hat{\Phi}(x_n)\}$. Then each test example was mapped into feature space and assigned the same label as $\text{argmin}_{x_i} \|\hat{\Phi}(x) - \hat{\Phi}(x_i)\|^2$). The total misclassification error was 2.2%, which can be directly compared with the misclassification error of the original classifiers of 2.85%. The CDM does better because it uses the training data explicitly *and* the information stored in the network to make a comparison, whereas the classifiers only use the information in the network. The learnt CDM was also used to do $k$-NN classification with $k > 1$. However this afforded no improvement. For example, the error of the 3-NN classifier was 2.54% and the error of the 20-NN classifier was 3.99%. This provides an indication that the CDM may not be the optimal distortion measure to use if $k$-NN classification ($k > 1$) is the aim.

In the second experiment $g$ was again used to do 1-NN classification on the test set, but this time all 3018 characters were distinguished. So in this case the learnt CDM was being asked to distinguish between 2618 characters that were treated as a single character when it was being trained. The misclassification error was a surprisingly low 7.5%. The 7.5% error compares favourably with the 4.8% error achieved on the same data by the CEDAR group, using a carefully selected feature set and a hand-tailored nearest-neighbour routine [5]. In our case the distance measure was *learnt* from raw-data input, and has not been the subject of any optimization or tweaking.

開 間 関 閣 問
案 業 素 実 常
普 群 勝 費 難
先 火 比 佐 死
暗 階 降 路 結
科 料 時 移 務

Figure 1: Six Kanji characters (first character in each row) and examples of their four nearest neighbours (remaining four characters in each row).

As a final, more qualitative assessment, the learnt CDM was used to compute the distance between every pair of testing examples, and then the distance between each pair of characters (an individual character being represented by a number of testing examples) was computed by averaging the distances between their constituent examples. The nearest neighbours of each character were then calculated. With this measure, every character turned out to be its own nearest neighbour, and in many cases the next-nearest neighbours bore a strong subjective similarity to the original. Some representative examples are shown in Figure 1.

## 6 CONCLUSION

We have shown how the Canonical Distortion Measure (CDM) is the optimal distortion measure for 1-NN classification, and that for environments in which all the functions can be expressed as a linear combination of a fixed set of features, the Canonical Distortion Measure is squared Euclidean distance in feature space. A technique for learning the CDM was presented and PAC-like bounds on the sample complexity required for good generalisation were proved.

Experimental results were presented in which the CDM for a Japanese OCR environment was learnt by first learning a common set of features for a subset of the character classifiers in the environment. The learnt CDM was then used as a distance measure in 1-NN neighbour classification, and performed remarkably well, both on the characters used to train it and on entirely novel characters.

## Footnotes

*The first author was supported in part by EPSRC grants #K70366 and #K70373

## References

[1] Jonathan Baxter. Learning Internal Representations. In *Proceedings of the Eighth International Conference on Computational Learning Theory*, pages 311–320. ACM Press, 1995.

[2] Jonathan Baxter. The Canonical Metric for Vector Quantisation. Technical Report NeuroColt Technical Report 047, Royal Holloway College, University of London, July 1995.

[3] Jonathan Baxter. The Canonical Distortion Measure for Vector Quantization and Function Approximation. In *Proceedings of the Fourteenth International Conference on Machine Learning*, July 1997. To Appear.

[4] W S Lee, P L Bartlett, and R C Williamson. Efficient agnostic learning of neural networks with bounded fan-in. *IEEE Transactions on Information Theory*, 1997.

[5] S.N. Srihari, T. Hong, and Z. Shi. Cherry Blossom: A System for Reading Unconstrained Handwritten Page Images. In *Symposium on Document Image Understanding Technology (SDIUT)*, 1997.
